# Multi-Level Active Prediction of Useful Image Annotations for Recognition

**Sudheendra Vijayanarasimhan and Kristen Grauman**
Department of Computer Sciences
University of Texas at Austin
{svnaras,grauman}@cs.utexas.edu

## Abstract

We introduce a framework for actively learning visual categories from a mixture of weakly and strongly labeled image examples. We propose to allow the category-learner to strategically choose what annotations it receives—based on both the expected reduction in uncertainty as well as the relative costs of obtaining each annotation. We construct a multiple-instance discriminative classifier based on the initial training data. Then all remaining unlabeled and weakly labeled examples are surveyed to actively determine which annotation ought to be requested next. After each request, the current classifier is incrementally updated. Unlike previous work, our approach accounts for the fact that the optimal use of manual annotation may call for a combination of labels at multiple levels of granularity (e.g., a full segmentation on some images and a present/absent flag on others). As a result, it is possible to learn more accurate category models with a lower total expenditure of manual annotation effort.

## 1  Introduction

Visual category recognition is a vital thread in computer vision research. The recognition problem remains challenging because of the wide variation in appearance a single class typically exhibits, as well as differences in viewpoint, illumination, and clutter. Methods are usually most reliable when good training sets are available, i.e., when labeled image examples are provided for each class, and where those training examples are adequately representative of the distribution to be encountered at test time. The extent of an image labeling can range from a flag telling whether the object of interest is present or absent, to a full segmentation specifying the object boundary. In practice, accuracy often improves with larger quantities of training examples and/or more elaborate annotations.

Unfortunately, substantial human effort is required to gather such training sets, making it unclear how the traditional protocol for visual category learning can truly scale. Recent work has begun to explore ways to mitigate the burden of supervision [1–8]. While the results are encouraging, existing techniques fail to address two key insights about low-supervision recognition: 1) the division of labor between the machine learner and the human labelers ought to respect any cues regarding which annotations would be easy (or hard) for either party to provide, and 2) to use a fixed amount of manual effort most effectively may call for a combination of annotations at multiple levels (e.g., a full segmentation on some images and a present/absent flag on others). Humans ought to be responsible for answering the hardest questions, while pattern recognition techniques ought to absorb and propagate that information and answer the easier ones. Meanwhile, the learning algorithm must be able to accommodate the multiple levels of granularity that may occur in provided image annotations, and to compute which item *at which of those levels* appears to be most fruitful to have labeled next (see Figure 1).

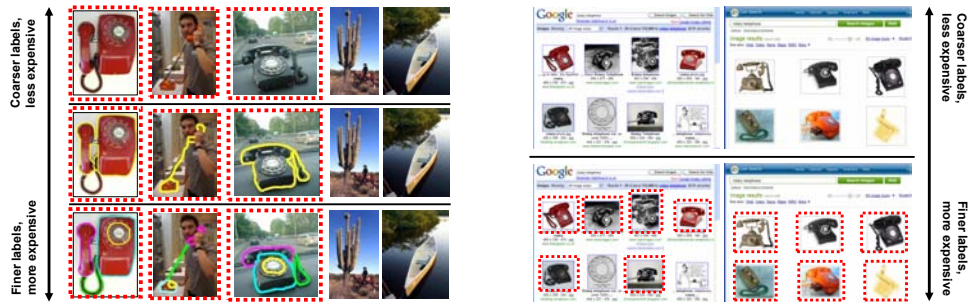

**Fig. 1.** Useful image annotations can occur at multiple levels of granularity. **Left:** For example, a learner may only know whether the image contains a particular object or not (top row, dotted boxes denote object is present), or it may also have segmented foregrounds (middle row), or it may have detailed outlines of object parts (bottom row). **Right:** In another scenario, groups of images for a given class are collected with keyword-based Web search. The learner may only be given the noisy groups and told that each includes at least one instance of the specified class (top), or, for some groups, the individual example images may be labeled as positive or negative (bottom). We propose an active learning paradigm that directs manual annotation effort to the most informative examples *and* levels.

To address this challenge, we propose a method that actively targets the learner's requests for supervision so as to maximize the expected benefit to the category models. Our method constructs an initial classifier from limited labeled data, and then considers all remaining unlabeled and weakly labeled examples to determine what annotation seems most informative to obtain. Since the varying levels of annotation demand varying degrees of manual effort, our active selection process weighs the value of the information gain against the cost of actually obtaining any given annotation. After each request, the current classifier is incrementally updated, and the process repeats.

Our approach accounts for the fact that image annotations can exist at multiple levels of granularity: both the classifier and active selection objectives are formulated to accommodate dual-layer labels. To achieve this duality for the classifier, we express the problem in the *multiple instance learning* (MIL) setting [9], where training examples are specified as bags of the finer granularity instances, and positive bags may contain an arbitrary number of negatives. To achieve the duality for the active selection, we design a decision-theoretic criterion that balances the variable costs associated with each type of annotation with the expected gain in information. Essentially this allows the learner to automatically predict when the extra effort of a more precise annotation is warranted.

The main contribution of this work is a unified framework to actively learn categories from a mixture of weakly and strongly labeled examples. We are the first to identify and address the problem of active visual category learning with multi-level annotations. In our experiments we demonstrate two applications of the framework for visual learning (as highlighted in Figure 1). Not only does our active strategy learn more quickly than a random selection baseline, but for a fixed amount of manual resources, it yields more accurate models than conventional single-layer active selection strategies.

## 2  Related Work

The recognition community is well-aware of the expense of requiring well-annotated image datasets. Recent methods have shown the possibility of learning visual patterns from unlabeled [3, 2] image collections, while other techniques aim to share or re-use knowledge across categories [10, 4]. Several authors have successfully leveraged the free but noisy images on the Web [5, 6, 11]. Using weakly labeled images to learn categories was proposed in [1], and several researchers have shown that MIL can accommodate the weak or noisy supervision often available for image data [11–14]. Working in the other direction, some research seeks to facilitate the manual labor of image annotation, tempting users with games or nice datasets [7, 8].

However, when faced with a distribution of unlabeled images, almost all existing methods for visual category learning are essentially passive, selecting points at random to label. Active learning strategies introduced in the machine learning literature generally select points so as to minimize the model entropy or reduce classification error (e.g., [15, 16]). Decision-theoretic measures for traditional (single-instance) learning have been explored in [17, 18], where they were applied to classify synthetic data and voicemail. Our active selection procedure is in part inspired by this work, as it

also seeks to balance the cost and utility tradeoff. Recent work has considered active learning with Gaussian Process classifiers [19], and relevance feedback for video annotations [20].

In contrast, we show how to form active multiple-instance learners, where constraints or labels must be sought at multiple levels of granularity. Further, we introduce the notion of predicting when to "invest" the labor of more expensive image annotations so as to ultimately yield bigger benefits to the classifier. Unlike any previous work, our method continually guides the annotation process to the appropriate level of supervision. While an active criterion for instance-level queries is suggested in [21] and applied within an MI learner, it cannot actively select positive bags or unlabeled bags, and does not consider the cost of obtaining the labels requested. In contrast, we formulate a general selection function that handles the full MIL paradigm and adapts according to the label costs. Experiments show this functionality to be critical for efficient learning from few images.

## 3 Approach

The goal of this work is to learn to recognize an object or category with minimal human intervention. The key idea is to actively determine which annotations a user should be asked to provide, and in what order. We consider image collections consisting of a variety of supervisory information: some images are labeled as containing the category of interest (or not), some have both a class label and a foreground segmentation, while others have no annotations at all. We derive an active learning criterion function that predicts how informative further annotation on any particular unlabeled image or region would be, while accounting for the variable expense associated with different annotation types. As long as the information expected from further annotations outweighs the cost of obtaining them, our algorithm will request the next valuable label, re-train the classifier, and repeat.

In the following we outline the MIL paradigm and discuss its applicability for two important image classification scenarios. Then, we describe our decision-theoretic approach to actively request useful annotations. Finally, we discuss how to attribute costs and risks for multi-level annotations.

### 3.1 Multiple-Instance Visual Category Learning

Traditional binary supervised classification assumes the learner is provided a collection of labeled data patterns, and must learn a function to predict labels on new instances. However, the fact that image annotations can exist at multiple levels of granularity demands a learning algorithm that can encode any known labels at the levels they occur, and so MIL [9] is more applicable. In MIL, the learner is instead provided with *sets* (bags) of patterns rather than individual patterns, and is only told that at least one member of any *positive bag* is truly positive, while every member of any *negative bag* is guaranteed to be negative. The goal of MIL is to induce the function that will accurately label individual instances such as the ones within the training bags.

MIL is well-suited for the following two image classification scenarios:

- Training images are labeled as to whether they contain the category of interest, but they also contain other objects and background clutter. Every image is represented by a bag of regions, each of which is characterized by its color, texture, shape, etc. [12, 13]. For positive bags, at least one of the regions contains the object of interest. The goal is to predict when new image regions contain the object—that is, to learn to label regions as foreground or background.

- The keyword associated with a category is used to download groups of images from multiple search engines in multiple languages. Each downloaded group is a bag, and the images within it are instances [11]. For each positive bag, at least one image actually contains the object of interest, while many others may be irrelevant. The goal is to predict the presence or absence of the category in new images.

In both cases, an instance-level decision is desirable, but bag-level labels are easier to obtain. While it has been established that MIL is valuable in such cases, previous methods do not consider how to determine what labels would be most beneficial to obtain.

We integrate our active selection method with the SVM-based MIL approach given in [22], which uses a Normalized Set Kernel (NSK) to describe bags based on the average representation of instances within them. Following [23], we use the NSK mapping for positive bags only; all instances in a negative bag are treated individually as negative. We chose this classifier since it performs well in practice [24] and allows incremental updates [25]; further, by virtue of being a kernel-based algorithm, it gives us flexibility in our choices of features and kernels. However, alternative MIL techniques that provide probabilitistic outputs could easily be swapped in (e.g. [26, 24, 23]).

## 3.2 Multi-Level Active Selection of Image Annotations

Given the current MIL classifier, our objective is to select what annotation should be requested next. Whereas active selection criteria for traditional supervised classifiers need only identify the best instance to label next, in the MIL domain we have a more complex choice. There are three possible types of request: the system can ask for a label on an instance, a label on an unlabeled bag, or for a joint labeling of all instances within a positive bag. So, we must design a selection criterion that simultaneously determines which type of annotation to request, and for which example to request it. Adding to the challenge, the selection process must also account for the variable costs associated with each level of annotation (e.g., it will take the annotator less time to detect whether the class of interest is present or not, while a full segmentation will be more expensive).

We extend the *value of information* (VOI) strategy proposed in [18] to enable active MIL selection, and derive a generalized value function that can accept both instances and bags. This allows us to predict the information gain in a joint labeling of multiple instances at once, and thereby actively choose when it is worthwhile to expend more or less manual effort in the training process. Our method continually re-evaluates the expected significance of knowing more about any unlabeled or partially labeled example, as quantified by the predicted reduction in misclassification risk plus the cost of obtaining the label.

We consider a collection of unlabeled data $\mathcal{X}_U$, and labeled data $\mathcal{X}_L$ composed of a set of positive bags $\mathcal{X}_p$ and a set of negative instances $\tilde{\mathcal{X}}_n$. Recall that positively labeled bags contain instances whose labels are unknown, since they contain an unknown mix of positive and negative instances. Let $r_p$ denote the user-specified risk associated with misclassifying a positive example as negative, and $r_n$ denote the risk of misclassifying a negative. The risk associated with the labeled data is:

$$Risk(\mathcal{X}_L) = \sum_{X_i \in \mathcal{X}_p} r_p(1 - p(X_i)) + \sum_{x_i \in \tilde{\mathcal{X}}_n} r_n p(x_i), \qquad (1)$$

where $x_i$ denotes an instance and $X_i$ denotes a bag. Here $p(x)$ denotes the probability that a given input is classified as positive: $p(x) = \Pr(\text{sgn}(w\phi(x) + b) = +1|x)$ for the SVM hyperplane parameters $w$ and $b$. We compute these values using the mapping suggested in [27], which essentially fits a sigmoid to map the SVM outputs to posterior probabilities. Note that here a positive bag $X_i$ is first transformed according to the NSK before computing its probability. The corresponding risk for unlabeled data is:

$$Risk(\mathcal{X}_U) = \sum_{x_i \in \mathcal{X}_U} r_p(1 - p(x_i)) \Pr(y_i = +1|x_i) \;+\; r_n p(x_i)(1 - \Pr(y_i = +1|x_i)), \qquad (2)$$

where $y_i$ is the true label for unlabeled example $x_i$. The value of $\Pr(y = +1|x)$ is not directly computable for unlabeled data; following [18], we approximate it as $\Pr(y = +1|x) \approx p(x)$. This simplifies the risk for the unlabeled data to: $Risk(\mathcal{X}_U) = \sum_{x_i \in \mathcal{X}_U} (r_p + r_n)(1 - p(x_i))p(x_i)$, where again we transform unlabeled bags according to the NSK before computing the posterior.

The total cost $T(\mathcal{X}_L, \mathcal{X}_U)$ associated with the data is the total misclassification risk, plus the cost of obtaining all labeled data thus far:

$$T(\mathcal{X}_L, \mathcal{X}_U) = Risk(\mathcal{X}_L) + Risk(\mathcal{X}_U) + \sum_{X_i \in \mathcal{X}_p} \mathcal{C}(X_i) + \sum_{x_i \in \tilde{\mathcal{X}}_n} \mathcal{C}(x_i), \qquad (3)$$

where the function $\mathcal{C}(\cdot)$ returns the cost of obtaining an annotation for its input, and will be defined in more detail below.

To measure the expected utility of obtaining any particular new annotation, we want to predict the *change* in total cost that would result from its addition to $\mathcal{X}_L$. Thus, the value of obtaining an annotation for input $\mathbf{z}$ is:

$$VOI(\mathbf{z}) = T(\mathcal{X}_L, \mathcal{X}_U) - T\left(\mathcal{X}_L \cup \mathbf{z}^{(t)}, \mathcal{X}_U \smallsetminus \mathbf{z}\right) \qquad (4)$$

$$= Risk(\mathcal{X}_L) + Risk(\mathcal{X}_U) - \left(Risk\left(\mathcal{X}_L \cup \mathbf{z}^{(t)}\right) + Risk\left(\mathcal{X}_U \smallsetminus \mathbf{z}\right)\right) - \mathcal{C}(\mathbf{z}),$$

where $\mathbf{z}^{(t)}$ denotes that the input $\mathbf{z}$ has been merged into the labeled set with its true label $t$, and $\mathcal{X}_U \smallsetminus \mathbf{z}$ denotes that it has been removed from the set of unlabeled data. If the VOI is high for a

given input, then the total cost would be decreased by adding its annotation; similarly, low values indicate minor gains, and negative values indicate an annotation that costs more to obtain than it is worth. Thus at each iteration, the active learner surveys all remaining unlabeled and weakly labeled examples, computes their VOI, and requests the label for the example with the maximal value.

However, there are two important remaining technical issues. First, for this to be useful we must be able to estimate the empirical risk for inputs before their labels are known. Secondly, for active selection to proceed at multiple levels, the VOI must act as an overloaded function: we need to be able to evaluate the VOI when $\mathbf{z}$ is an unlabeled instance *or* an unlabeled bag *or* a weakly labeled example, i.e., a positive bag containing an unknown number of negative instances.

To estimate the total risk induced by incorporating a newly annotated example $\mathbf{z}$ into $\mathcal{X}_L$ before actually obtaining its true label $t$, we estimate the updated risk term with its expected value: $Risk(\mathcal{X}_L \cup \mathbf{z}^{(t)}) + Risk(\mathcal{X}_U \setminus \mathbf{z}) \approx E[Risk(\mathcal{X}_L \cup \mathbf{z}^{(t)}) + Risk(\mathcal{X}_U \setminus \mathbf{z})] = \mathbb{E}$, where $\mathbb{E}$ is shorthand for the expected value expression preceding it. If $\mathbf{z}$ is an unlabeled instance, then computing the expectation is straightforward:

$$\mathbb{E} = \sum_{l \in \mathbb{L}} \Big( Risk(\mathcal{X}_L \cup \mathbf{z}^{(l)}) + Risk(\mathcal{X}_U \setminus \mathbf{z}) \Big) \Pr(\mathrm{sgn}(w\phi(\mathbf{z}) + b) = l | \mathbf{z}), \qquad (5)$$

where $\mathbb{L} = \{+1, -1\}$ is the set of all possible label assignments for $\mathbf{z}$. The value $\Pr(\mathrm{sgn}(w\phi(\mathbf{z}) + b) = l | \mathbf{z})$ is obtained by evaluating the current classifier on $\mathbf{z}$ and mapping the output to the associated posterior, and risk is computed based on the (temporarily) modified classifier with $\mathbf{z}^{(l)}$ inserted into the labeled set. Similarly, if $\mathbf{z}$ is an unlabeled bag, the label assignment can only be positive or negative, and we compute the probability of either label via the NSK mapping.

If $\mathbf{z}$ is a positive bag containing $M = |\mathbf{z}|$ instances, however, there are $2^M$ possible labelings: $\mathbb{L} = \{+1, -1\}^M$. For even moderately sized bags, this makes a direct computation of the expectation impractical. Instead, we use Gibbs sampling to draw samples of the label assignment from the joint distribution over the $M$ instances' descriptors. Let $\mathbf{z} = \{z_1, \dots, z_M\}$ be the positive bag's instances, and let $\mathbf{z}^{(\mathbf{a})} = \left\{ (z_1^{(a_1)}), \dots, (z_M^{(a_M)}) \right\}$ denote the label assignment we wish to sample, with $a_j \in \{+1, -1\}$. To sample from the conditional distribution of one instance's label given the rest—the basic procedure required by Gibbs sampling—we re-train the MIL classifier with the given labels added, and then draw the remaining label according to $a_j \sim \Pr(\mathrm{sgn}(w\phi(z_j) + b) = +1 | z_j)$, where $z_j$ denotes the one instance currently under consideration. For positive bag $\mathbf{z}$, the expected total risk is then the average risk computed over all $S$ generated samples:

$$\mathbb{E} = \frac{1}{S} \sum_{k=1}^{S} \Big( Risk(\{\mathcal{X}_L \setminus \mathbf{z}\} \cup \{z_1^{(a_1)_k}, \dots, z_M^{(a_M)_k}\}) + Risk(\mathcal{X}_U \setminus \{z_1, z_2, ..., z_M\}) \Big), \quad (6)$$

where $k$ indexes the $S$ samples. To compute the risk on $\mathcal{X}_L$ for each fixed sample we simply remove the weakly labeled positive bag $\mathbf{z}$, and insert its instances as labeled positives and negatives, as dictated by the sample's label assignment. Computing the VOI values for all unlabeled data, especially for the positive bags, requires repeatedly solving the classifier objective function with slightly different inputs; to make this manageable we employ incremental SVM updates [25].

To complete our active selection function, we must define the cost function $\mathcal{C}(\mathbf{z})$, which maps an input to the amount of effort required to annotate it. This function is problem-dependent. In the visual categorization scenarios we have set forth, we define the cost function in terms of the type of annotation required for the input $\mathbf{z}$; we charge equal cost to label an instance or an unlabeled bag, and proportionally greater cost to label all instances in a positive bag, as determined empirically with labeling experiments with human users. This reflects that outlining an object contour is more expensive than naming an object, or sorting through an entire page of Web search returns is more work than labeling just one.

We can now actively select which examples and what type of annotation to request, so as to maximize the expected benefit to the category model relative to the manual effort expended. After each annotation is added and the classifier is revised accordingly, the VOI is evaluated on the remaining unlabeled and weakly labeled data in order to choose the next annotation. This process repeats either until the available amount of manual resources is exhausted, or, alternatively, until the maximum VOI is negative, indicating further annotations are not worth the effort.

## 4 Results

In this section we demonstrate our approach to actively learn visual categories. We test with two distinct publicly available datasets that illustrate the two learning scenarios above: (1) the SIVAL dataset[1] of 25 objects in cluttered backgrounds, and (2) a Google dataset ([5]) of seven categories downloaded from the Web. In both, the classification task is to say whether each unseen image contains the object of interest or not. We provide comparisons with single-level active learning (with both the method of [21], and where the same VOI function is used but is restricted to actively label only instances), as well as passive learning. For the passive baseline, we consider random selections from amongst both single-level and multi-level annotations, in order to verify that our approach does not simply benefit from having access to more informative possible labels. [2]

To determine how much more labeling a positive bag costs relative to labeling an instance, we performed user studies for both of the scenarios evaluated. For the first scenario, users were shown oversegmented images and had to click on all the segments belonging to the object of interest. In the second, users were shown a page of downloaded Web images and had to click on only those images containing the object of interest. For both datasets, their baseline task was to provide a present/absent flag on the images. For segmentation, obtaining labels on all positive segments took users on average four times as much time as setting a flag. For the Web images, it took 6.3 times as long to identify all positives within bags of 25 noisy images. Thus we set the cost of labeling a positive bag to 4 and 6.3 for the SIVAL and Google data, respectively. These values agree with the average sparsity of the two datasets: the Google set contains about 30% true positive images while the SIVAL set contains 10% positive segments per image. The users who took part in the experiment were untrained but still produced consistent results.

### 4.1 Actively Learning Visual Objects and their Foreground Regions from Cluttered Images

The SIVAL dataset [21] contains 1500 images, each labeled with one of 25 class labels. The cluttered images contain objects in a variety of positions, orientations, locations, and lighting conditions. The images have been oversegmented into about 30 regions (instances) each, each of which is represented by a 30-d feature describing its color and texture. Thus each image is a bag containing both positive and negative instances (segments). Labels on the training data specify whether the object of interest is present or not, but the segments themselves are unlabeled (though the dataset does provide ground truth segment labels for evaluation purposes).

The initial training set is comprised of 10 positive and 10 negative images per class, selected at random. Our active learning method must choose its queries from among 10 positive bags (complete segmentations), 300 unlabeled instances (individual segments), and about 150 unlabeled bags (present/absent flag on the image). We use a quadratic kernel with a coefficient of $10^{-6}$, and average results over five random training partitions.

Figure 2(a) shows representative (best and worst) learning curves for our method and the three baselines, all of which use the same MIL classifier (NSK-SVM). Note that the curves are plotted against the cumulative *cost* of obtaining labels—as opposed to the number of queried instances—since our algorithm may choose a sequence of queries with non-uniform cost. All methods are given a fixed amount of manual effort (40 cost units) and are allowed to make a sequence of choices until that cost is used up. Recall that a cost of 40 could correspond, for example, to obtaining labels on $\frac{40}{1} = 40$ instances or $\frac{40}{4} = 10$ positive bags, or some mixture thereof. Figure 2(b) summarizes the learning curves for all categories, in terms of the average improvement at a fixed point midway through the active learning phase.

All four methods steadily improve upon the initial classifier, but at different rates with respect to the cost. (All methods fail to do better than chance on the 'dirty glove' class, which we attribute to the lack of distinctive texture or color on that object.) In general, a steeper learning curve indicates that a method is learning most effectively from the supplied labels. Our multi-level approach shows the most significant gains at a lower cost, meaning that it is best suited for building accurate classifiers with minimal manual effort on this dataset. As we would expect, single-level active selections are better than random, but still fall short of our multi-level approach. This is because single-level active selection can only make a sequence of greedy choices while our approach can jointly select bags of instances to query. Interestingly, multi- and single-level random selections perform quite similarly

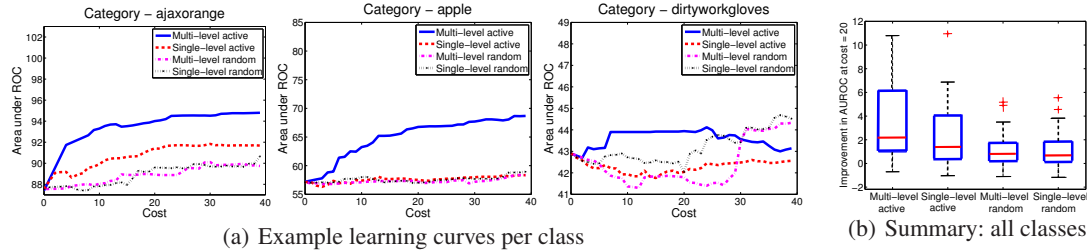

(a) Example learning curves per class     (b) Summary: all classes

**Fig. 2.** Results on the SIVAL dataset. **(a)** Sample learning curves per class, each averaged over five trials. First two are best examples, last is worst. **(b)** Summary of the average improvement over all categories after half of the annotation cost is used. For the same amount of annotation cost, our multi-level approach learns more quickly than both traditional single-level active selection as well as both forms of random selection.

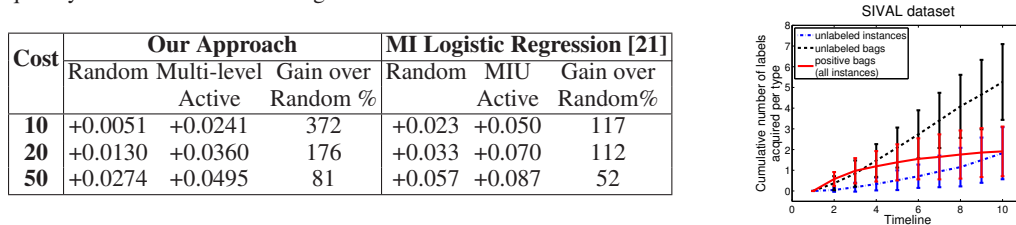

| Cost | Our Approach | | | MI Logistic Regression [21] | | |
|---|---|---|---|---|---|---|
| | Random | Multi-level Active | Gain over Random % | Random | MIU Active | Gain over Random% |
| 10 | +0.0051 | +0.0241 | 372 | +0.023 | +0.050 | 117 |
| 20 | +0.0130 | +0.0360 | 176 | +0.033 | +0.070 | 112 |
| 50 | +0.0274 | +0.0495 | 81 | +0.057 | +0.087 | 52 |

**Fig. 3. Left:** Comparison with [21] on the SIVAL data, as measured by the average improvement in the AUROC over the initial model for increasing labeling cost values. **Right:** The cumulative number of labels acquired for each type with increasing number of queries. Our method tends to request complete segmentations or image labels early on, followed by queries on unlabeled segments later on.

on this dataset (see boxplots in (b)), which indicates that having more informative labels alone does not directly lead to better classifiers unless the right instances are queried.

The table in Figure 3 compares our results to those reported in [21], in which the authors train an initial classifier with *multiple-instance logistic regression*, and then use the MI Uncertainty (MIU) to actively choose instances to label. Following [21], we report the average gains in the AUROC over all categories at fixed points on the learning curve, averaging results over 20 trials and with the same initial training set of 20 positive and negative images. Since the accuracy of the base classifiers used by the two methods varies, it is difficult to directly compare the gains in the AUROC. The NSK-SVM we use consistently outperforms the logistic regression approach using only the initial training set; even before active learning our average accuracy is 68.84, compared to 52.21 in [21]. Therefore, to aid in comparison, we also report the percentage gain relative to random selection, for both classifiers. The results show that our approach yields much stronger relative improvements, again illustrating the value of allowing active choices at multiple levels. For both methods, the percent gains decrease with increasing cost; this makes sense, since eventually (for enough manual effort) a passive learner can begin to catch up to an active learner.

## 4.2 Actively Learning Visual Categories from Web Images

Next we evaluate the scenario where each positive bag is a collection of images, among which only a portion are actually positive instances for the class of interest. Bags are formed from the Google-downloaded images provided in [5]. This set contains on average 600 examples for each of the seven categories. Naturally, the number of true positives for each class are sparse: on average 30% contain a "good" view of the class of interest, 20% are of "ok" quality (occlusions, noise, cartoons, etc.), and 50% are "junk". Previous methods have shown how to learn from noisy Web images, with results rivaling state-of-the-art supervised techniques [11, 5, 6]. We show how to boost accuracy with these types of learners while leveraging minimal manual annotation effort.

To re-use the publicly available dataset from [5], we randomly group Google images into bags of size 25 to simulate multiple searches as in [11], yielding about 30 bags per category. We randomly select 10 positive and 10 negative bags (from all other categories) to serve as the initial training data for each class. The rest of the positive bags of a class are used to construct the test sets. All results are averaged over five random partitions. We represent each image as a bag of "visual words", and compare examples with a linear kernel. Our method makes active queries among 10 positive bags (complete labels) and about 250 unlabeled instances (images). There are no unlabeled bags in this scenario, since every downloaded batch is associated with a keyword.

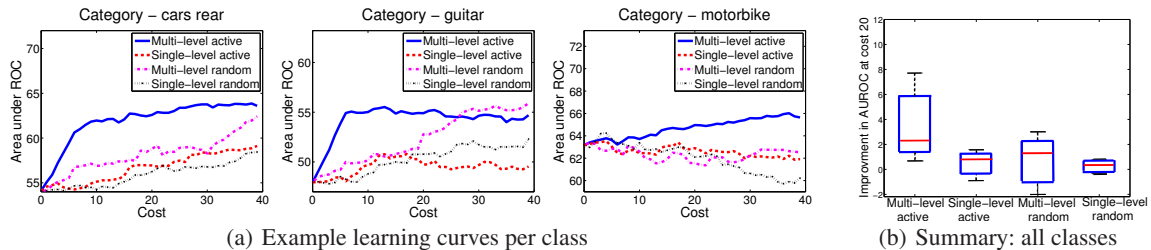

(a) Example learning curves per class       (b) Summary: all classes

**Fig. 4.** Results on the Google dataset, in the same format as Figure 2. Our multi-level active approach outperforms both random selection strategies and traditional single-level active selection.

Figure 4 shows the learning curves and a summary of our active learner's performance. Our multi-level approach again shows more significant gains at a lower cost relative to all baselines, improving accuracy with as few as ten labeled instances. On this dataset, random selection with multi-level annotations actually outperforms random selection on single-level annotations (see the boxplots). We attribute this to the distribution of bags/instances: on average more positive bags were randomly chosen, and each addition led to a larger increase in the AUROC.

## 5 Conclusions and Future Work

Our approach addresses a new problem: how to actively choose not only which instance to label, but also what type of image annotation to acquire in a cost-effective way. Our method is general enough to accept other types of annotations or classifiers, as long as the cost and risk functions can be appropriately defined. Comparisons with passive learning methods and single-level active learning show that our multi-level method is better-suited for building classifiers with minimal human intervention. In future work, we will consider look-ahead scenarios with more far-sighted choices. We are also pursuing ways to alleviate the VOI computation cost, which as implemented involves processing all unlabeled data prior to making a decision. Finally, we hope to incorporate our approach within an existing system with many real users, like Labelme [8].

## Footnotes

[1] http://www.cs.wustl.edu/accio/

[2] See [28] for further implementation details, image examples, and learning curves on all classes.

## References

[1] Weber, M., Welling, M., Perona, P.: Unsupervised Learning of Models for Recognition. In: ECCV. (2000)
[2] Sivic, J., Russell, B., Efros, A., Zisserman, A., Freeman, W.: Discovering Object Categories in Image Collections. In: ICCV. (2005)
[3] Quelhas, P., Monay, F., Odobez, J.M., Gatica-Perez, D., Tuytelaars, T., VanGool, L.: Modeling Scenes with Local Descriptors and Latent Aspects. In: ICCV. (2005)
[4] Bart, E., Ullman, S.: Cross-Generalization: Learning Novel Classes from a Single Example by Feature Replacement. In: CVPR. (2005)
[5] Fergus, R., Fei-Fei, L., Perona, P., Zisserman, A.: Learning Object Categories from Google's Image Search. In: ICCV. (2005)
[6] Li, L., Wang, G., Fei-Fei, L.: Optimol: Automatic Online Picture Collection via Incremental Model Learning. In: CVPR. (2007)
[7] von Ahn, L., Dabbish, L.: Labeling Images with a Computer Game. In: CHI. (2004)
[8] Russell, B., Torralba, A., Murphy, K., Freeman, W.: Labelme: a Database and Web-Based Tool for Image Annotation. TR, MIT (2005)
[9] Dietterich, T., Lathrop, R., Lozano-Perez, T.: Solving the Multiple Instance Problem with Axis-Parallel Rectangles. Artificial Intelligence **89** (1997) 31–71
[10] Murphy, K., Torralba, A., Freeman, W.: Using the Forest to See the Trees:a Graphical Model Relating Features, Objects and Scenes. In: NIPS. (2003)
[11] Vijayanarasimhan, S., Grauman, K.: Keywords to Visual Categories: Multiple-Instance Learning for Weakly Supervised Object Categorization. In: CVPR. (2008)
[12] Maron, O., Ratan, A.: Multiple-Instance Learning for Natural Scene Classification. In: ICML. (1998)
[13] Yang, C., Lozano-Perez, T.: Image Database Retrieval with Multiple-Instance Learning Techniques. In: ICDE. (2000)
[14] Viola, P., Platt, J., Zhang, C.: Multiple Instance Boosting for Object Detection. In: NIPS. (2005)
[15] Freund, Y., Seung, H., Shamir, E., Tishby: Selective Sampling Using the Query by Committee Algorithm. Machine Learning **28** (1997)
[16] Tong, S., Koller, D.: Support Vector Machine Active Learning with Applications to Text Classification. In: ICML. (2000)
[17] Lindenbaum, M., Markovitch, S., Rusakov, D.: Selective Sampling for Nearest Neighbor Classifiers. Machine Learning **54** (2004)
[18] Kapoor, A., Horvitz, E., Basu, S.: Selective Supervision: Guiding Supervised Learning with Decision-Theoretic Active Learning. In: IJCAI. (2007)
[19] Kapoor, A., Grauman, K., Urtasun, R., Darrell, T.: Active Learning with Gaussian Processes for Object Categorization. In: ICCV. (2007)
[20] Yan, R., Yang, J., Hauptmann, A.: Automatically Labeling Video Data using Multi-Class Active Learning. In: ICCV. (2003)
[21] Settles, B., Craven, M., Ray, S.: Multiple-Instance Active Learning. In: NIPS. (2008)
[22] Gartner, T., Flach, P., Kowalczyk, A., Smola, A.: Multi-Instance Kernels. In: ICML. (2002)
[23] Bunescu, R., Mooney, R.: Multiple Instance Learning for Sparse Positive Bags. In: ICML. (2007)
[24] Ray, S., Craven, M.: Supervised v. Multiple Instance Learning: An Empirical Comparison. In: ICML. (2005)
[25] Cauwenberghs, G., Poggio, T.: Incremental and Decremental Support Vector Machine Learning. In: NIPS. (2000)
[26] Andrews, S., Tsochantaridis, I., Hofmann, T.: Support Vector Machines for Multiple-Instance Learning. In: NIPS. (2002)
[27] Platt, J.: Probabilistic Outputs for Support Vector Machines and Comparisons to Regularized Likelihood Methods. In: Advances in Large Margin Classifiers. MIT Press (1999)
[28] Vijayanarasimhan, S., Grauman, K.: Multi-level Active Prediction of Useful Image Annotations for Recognition. Technical Report UT-AI-TR-08-2, University of Texas at Austin (2008)
